# Universal Kernels on Non-Standard Input Spaces

**Andreas Christmann**
University of Bayreuth
Department of Mathematics
D-95440 Bayreuth
andreas.christmann@uni-bayreuth.de

**Ingo Steinwart**
University of Stuttgart
Department of Mathematics
D-70569 Stuttgart
ingo.steinwart@mathematik.uni-stuttgart.de

## Abstract

During the last years support vector machines (SVMs) have been successfully applied in situations where the input space $X$ is not necessarily a subset of $\mathbb{R}^d$. Examples include SVMs for the analysis of histograms or colored images, SVMs for text classification and web mining, and SVMs for applications from computational biology using, e.g., kernels for trees and graphs. Moreover, SVMs are known to be consistent to the Bayes risk, if either the input space is a complete separable metric space and the reproducing kernel Hilbert space (RKHS) $H \subset L_p(\mathrm{P}_X)$ is dense, or if the SVM uses a universal kernel $k$. So far, however, there are no kernels of practical interest known that satisfy these assumptions, if $X \not\subset \mathbb{R}^d$. We close this gap by providing a general technique based on Taylor-type kernels to explicitly construct universal kernels on compact metric spaces which are not subset of $\mathbb{R}^d$. We apply this technique for the following special cases: universal kernels on the set of probability measures, universal kernels based on Fourier transforms, and universal kernels for signal processing.

## 1 Introduction

For more than a decade, kernel methods such as support vector machines (SVMs) have belonged to the most successful learning methods. Besides several other nice features, one key argument for using SVMs has been the so-called "kernel trick" [22], which decouples the SVM optimization problem from the domain of the samples, thus making it possible to use SVMs on virtually any input space $X$. This flexibility is in strong contrast to more classical learning methods from both machine learning and non-parametric statistics, which almost always require input spaces $X \subset \mathbb{R}^d$. As a result, kernel methods have been successfully used in various application areas that were previously infeasible for machine learning methods. The following, by no means exhaustive, list illustrates this:

- SVMs processing probability measures, e.g. histograms, as input samples have been used to analyze histogram data such as colored images, see [5, 11, 14, 12, 27, 29], and also [17] for non-extensive information theoretic kernels on measures.
- SVMs for text classification and web mining [15, 12, 16],
- SVMs with kernels from computational biology, e.g. kernels for trees and graphs [23].

In addition, several extensions or generalizations of kernel-methods have been considered, see e.g. [13, 26, 9, 16, 7, 8, 4]. Besides their practical success, SVMs nowadays also possess a rich

statistical theory, which provides various learning guarantees, see [31] for a recent account. Interestingly, in this analysis, the kernel and its reproducing kernel Hilbert space (RKHS) make it possible to completely decouple the *statistical* analysis of SVMs from the input space $X$. For example, if one uses the hinge loss and a bounded measurable kernel whose RKHS $H$ is separable and dense in $L_1(\mu)$ for all distributions $\mu$ on $X$, then [31, Theorem 7.22] together with [31, Theorem 2.31] and the discussion on [31, page 267ff] shows that the corresponding SVM is universally classification consistent even without an entropy number assumption if one picks a sequence $(\lambda_n)$ of positive regularization parameters that satisfy $\lambda_n \to 0$ and $n\lambda_n/\ln n \to \infty$. In other words, independently of the input space $X$, the universal consistency of SVMs is well-understood modulo an approximation theoretical question, namely that of the denseness of $H$ in all $L_1(\mu)$.

For standard input spaces $X \subset \mathbb{R}^d$ and various classical kernels, this question of denseness has been positively answered. For example, for *compact* $X \subset \mathbb{R}^d$, [30] showed that, among a few others, the RKHSs of the Gaussian RBF kernels are *universal*, that is, they are dense in the space $C(X)$ of continuous functions $f : X \to \mathbb{R}$. With the help of a standard result from measure theory, see e.g. [1, Theorem 29.14], it is then easy to conclude that these RKHS are also dense in all $L_1(\mu)$ for which $\mu$ has a compact support. This key result has been extended in a couple of different directions: For example, [18] establishes universality for more classes of kernels on compact $X \subset \mathbb{R}^d$, whereas [32] shows the denseness of the Gaussian RKHSs in $L_1(\mu)$ for *all* distributions $\mu$ on $\mathbb{R}^d$. Finally, [7, 8, 28, 29] show that universal kernels are closely related to so-called *characteristic* kernels that can be used to distinguish distributions. In addition, all these papers contain sufficient or necessary conditions for universality of kernels on arbitrary compact metric spaces $X$, and [32] further shows that the compact metric spaces are exactly the compact topological spaces on which there exist universal spaces.

Unfortunately, however, it appears that neither the sufficient conditions for universality nor the proof of the existence of universal kernels can be used to *construct* universal kernels on compact metric spaces $X \not\subset \mathbb{R}^d$. In fact, to the best of our knowledge, no explicit example of such kernels has so far been presented. As a consequence, it seems fair to say that, beyond the $X \subset \mathbb{R}^d$-case, the theory of SVMs is incomplete, which is in contrast to the obvious practical success of SVMs for such input spaces $X$ as illustrated above.

*The goal of this paper is to close this gap by providing the first explicit and constructive examples of universal kernels that live on compact metric spaces $X \not\subset \mathbb{R}^d$.* To achieve this, our first step is to extend the definition of the Gaussian RBF kernels, or more generally, kernels that can be expressed by a Taylor series, from the Euclidean $\mathbb{R}^d$ to its infinite dimensional counter part, that is, the space $\ell_2$ of square summable sequences. Unfortunately, on the space $\ell_2$ we face new challenges due to its infinite dimensional nature. Indeed, the closed balls of $\ell_2$ are no longer (norm)-compact subsets of $\ell_2$ and hence we cannot expect universality on these balls. To address this issue, one may be tempted to use the weak*-topology on $\ell_2$, since in this topology the closed balls are both compact and metrizable, thus universal kernels do exist on them. However, the Taylor kernels do not belong to them, because –basically– the inner product $\langle \cdot, \cdot \rangle_{\ell_2}$ fails to be continuous with respect to the weak*-topology as the sequence of the standard orthonormal basis vectors show. To address this compactness issue we consider (norm)-compact subsets of $\ell_2$, only. Since the inner product of $\ell_2$ is continuous with respect to the norm by virtue of the Cauchy-Schwarz inequality, it turns out that the Taylor kernels are continuous with respect to the norm topology. Moreover, we will see that in this situation the Stone-Weierstraß-argument of [30] yields a variety of universal kernels including the infinite dimensional extensions of the Gaussian RBF kernels.

However, unlike the finite dimensional Euclidean spaces $\mathbb{R}^d$ and their compact subsets, the compact subsets of $\ell_2$ can be hardly viewed as somewhat natural examples of input spaces $X$. Therefore, we go one step further by considering compact metric spaces $X$ for which there exist a separable Hilbert space $\mathcal{H}$ and an injective and continuous map $\rho : X \to \mathcal{H}$. If, in this case, we fix an analytic function $K : \mathbb{R} \to \mathbb{R}$ that can be globally expressed by its Taylor series developed at zero and that has strictly positive Taylor coefficients, then $k(x, x') := K(\langle \rho(x), \rho(x') \rangle_{\mathcal{H}})$ defines a universal kernel on $X$ and the same is true for the analogous definition of Gaussian kernels. Although this situation may look at a first glance even more artificial than the $\ell_2$-case, it turns out that quite a few interesting explicit examples can be derived from this situation. Indeed, we will use this general result to present examples of Gaussian kernels defined on the set of distributions over some input space $\Omega$ and on certain sets of functions.

The paper has the following structure. Section 2 contains the main results and constructs examples for universal kernels based on our technique. In particular, we show how to construct universal kernels on sets of probability measures and on sets of functions, the latter being interesting for signal processing. Section 3 contains a short discussion and Section 4 gives the proofs of the main results.

## 2 Main result

A kernel $k$ on a set $X$ is a function $k : X \times X \to \mathbb{R}$ for which all matrices of the form $(k(x_i, x_j))_{i,j=1}^n$, $n \in \mathbb{N}$, $x_1, \ldots, x_n \in X$, are symmetric and positive semi-definite. Equivalently, $k$ is a kernel if and only there exists a Hilbert space $\tilde{H}$ and a map $\tilde{\Phi} : X \to \tilde{H}$ such that $k(x, x') = \langle \tilde{\Phi}(x), \tilde{\Phi}(x') \rangle_{\tilde{H}}$ for all $x, x' \in X$. While neither $\tilde{H}$ or $\tilde{\Phi}$ are uniquely determined, the so-called reproducing kernel Hilbert space (RKHS) of $k$, which is given by

$$H := \{ \langle v, \Phi(\cdot) \rangle_{\tilde{H}} : v \in \tilde{H} \}$$

and $\|f\|_H := \inf\{\|v\|_{\tilde{H}} : f = \langle v, \Phi(\cdot) \rangle_{\tilde{H}}\}$ *is* uniquely determined, see e.g. [31, Chapter 4.2]. For more information on kernels, we refer to [31, Chapter 4]. Moreover, for a compact metric space $(X, d)$, we write $C(X) := \{f : X \to \mathbb{R} \mid f \text{ continuous}\}$ for the space of continuous functions on $X$ and equip this space with the usual supremum norm $\|\cdot\|_\infty$. A kernel $k$ on $X$ is called *universal*, if $k$ is continuous and its RKHS $H$ is dense in $C(X)$. As mentioned before, this notion, which goes back to [30], plays a key role in the analysis of kernel-based learning methods. Let $r \in (0, \infty]$. The kernels we consider in this paper are constructed by functions $K : [-r, r] \to \mathbb{R}$ that can be expressed by its Taylor series, that is

$$K(t) = \sum_{n=0}^\infty a_n t^n, \qquad t \in [-r, r]. \tag{1}$$

For such functions [31, Lemma 4.8] showed that

$$k(x, x') := K(\langle x, x' \rangle_{\mathbb{R}^d}) = \sum_{n=0}^\infty a_n \langle x, x' \rangle_{\mathbb{R}^d}^n, \qquad x, x' \in \sqrt{r} B_{\mathbb{R}^d}, \tag{2}$$

defines a kernel on the closed ball $\sqrt{r} B_{\mathbb{R}^d} := \{x \in \mathbb{R}^d : \|x\|_2 \le \sqrt{r}\}$ with radius $\sqrt{r}$, whenever all Taylor coefficients $a_n$ are non-negative. Following [31], we call such kernels Taylor kernels. [30], see also [31, Lemma 4.57], showed that Taylor kernels are universal, if $a_n > 0$ for all $n \ge 0$, while [21] notes that strict positivity on certain subsets of indices $n$ suffices.

Obviously, the definition (2) of $k$ is still possible, if one replaces $\mathbb{R}^d$ by its infinite dimensional and separable counterpart $\ell_2 := \{(w_j)_{j \ge 1} : \|(w_j)\|_{\ell_2}^2 := \sum_{j \ge 1} w_j^2 < \infty\}$. Let us denote the closed unit ball in $\ell_2$ by $B_{\ell_2}$, or more generally, the closed unit ball of a Banach space $E$ by $B_E$, that is $B_E := \{v \in E : \|v\|_E \le 1\}$. Our first main result shows that this extension leads to a kernel, whose restrictions to compact subsets are universal, if $a_n > 0$ for all $n \in \mathbb{N}_0 := \mathbb{N} \cup \{0\}$.

**Theorem 2.1** *Let $K : [-r, r] \to \mathbb{R}$ be a function of the form (1). Then we have:*

i) *If $a_n \ge 0$ for all $n \ge 0$, then $k : \sqrt{r} B_{\ell_2} \times \sqrt{r} B_{\ell_2} \to \mathbb{R}$ is a kernel, where*

$$k(w, w') := K(\langle w, w' \rangle_{\ell_2}) = \sum_{n=0}^\infty a_n \langle w, w' \rangle_{\ell_2}^n, \qquad w, w' \in \sqrt{r} B_{\ell_2}. \tag{3}$$

ii) *If $a_n > 0$ for all $n \in \mathbb{N}_0$, then the restriction $k_{|W \times W} : W \times W \to \mathbb{R}$ of $k$ to an arbitrary compact set $W \subset \sqrt{r} B_{\ell_2}$ is universal.*

To consider a first explicit example, let $K := \exp : \mathbb{R} \to \mathbb{R}$ be the exponential function. Then $K$ clearly satisfies the assumptions of Theorem 2.1 for all $r > 0$, and hence the resulting *exponential* kernel is universal on every compact subset $W$ of $\ell_2$. Moreover, for $\sigma \in (0, \infty)$, the related Gaussian-type RBF kernel $k_\sigma : \ell_2 \times \ell_2 \to \mathbb{R}$ defined by

$$k_\sigma(w, w') := \exp(-\sigma^2 \|w - w'\|_{\ell_2}^2) = \frac{\exp(2\sigma^2 \langle w, w' \rangle_{\ell_2})}{\exp(\sigma^2 \|w\|_{\ell_2}^2) \exp(\sigma^2 \|w'\|_{\ell_2}^2)} \tag{4}$$

is also universal on every compact $W \subset \ell_2$, since modulo the scaling by $\sigma$ it is the normalized version of the exponential kernel, and thus it is universal by [31, Lemma 4.55].

Although we have achieved our first goal, namely explicit, constructive examples of universal kernels on $X \not\subset \mathbb{R}^d$, the result is so far not really satisfying. Indeed, unlike the finite dimensional Euclidean spaces $\mathbb{R}^d$, the infinite dimensional space $\ell_2$ rarely appears as the input space in real-world applications. The following second result can be used to address this issue.

**Theorem 2.2** *Let $X$ be a compact metric space and $\mathcal{H}$ be a separable Hilbert space such that there exists a continuous and injective map $\rho : X \to \mathcal{H}$. Furthermore, let $K : \mathbb{R} \to \mathbb{R}$ be a function of the form (1). Then the following statements hold:*

  *i) If $a_n \geq 0$ for all $n \in \mathbb{N}_0$, then $k : X \times X \to \mathbb{R}$ defines a kernel, where*

$$k(x, x') := K\big(\langle \rho(x), \rho(x')\rangle_{\mathcal{H}}\big) = \sum_{n=0}^{\infty} a_n \langle \rho(x), \rho(x')\rangle_{\mathcal{H}}^n, \qquad x, x' \in X. \quad (5)$$

  *ii) If $a_n > 0$ for all $n \in \mathbb{N}_0$, then $k$ is a universal kernel.*

  *iii) For $\sigma > 0$, the Gaussian-type RBF-kernel $k_\sigma : X \times X \to \mathbb{R}$ is a universal kernel, where*

$$k_\sigma(x, x') := \exp\big(-\sigma^2 \|\rho(x) - \rho(x')\|_{\mathcal{H}}^2\big), \qquad x, x' \in X. \quad (6)$$

It seems possible that the latter result for the Gaussian-type RBF kernel can be extended to other positive non-constant radial basis function kernels such as $k_\sigma(x, x') := \exp\big(-\sigma^2 \|\rho(x) - \rho(x')\|_{\mathcal{H}}\big)$ or the Student-type RBF kernels $k_\sigma(x, x') := \big(1 + \sigma^2 \|\rho(x) - \rho(x')\|_{\mathcal{H}}^2\big)^{-\alpha}$ for $\sigma^2 > 0$ and $\alpha \geq 1$. Indeed, [25] uses the fact that on $\mathbb{R}^d$ such kernels have an integral representation in terms of the Gaussian RBF kernels to show, see [25, Corollary 4.9], that these kernels inherit approximation properties such as universality from the Gaussian RBF kernels. We expect that the same arguments can be made for $\ell_2$ and then, in a second step, for the situation of Theorem 2.2.

Before we provide some examples of situations in which Theorem 2.2 can be used to define explicit universal kernels, we point to a technical detail of Theorem 2.2, which may be overseen, thus leading to wrong conclusions.

To this end, let $(X, d_X)$ be an arbitrary metric space, $\mathcal{H}$ be a separable Hilbert space and $\rho : X \to \mathcal{H}$ be an injective map. We write $V := \rho(X)$ and equip this space with the metric defined by $\mathcal{H}$. Thus, $\rho : X \to V$ is bijective by definition. Moreover, since $H$ is assumed to be separable, it is isometrically isomorphic to $\ell_2$, and hence there exists an isometric isomorphism $I : \mathcal{H} \to \ell_2$. We write $W := I(V)$ and equip this set with the metric defined by the norm of $\ell_2$. For a function $f : W \to \mathbb{R}$, we can then consider the following diagram

$$
\begin{array}{ccc}
(X, d_X) & \xrightarrow{\ f \circ I \circ \rho\ } & (\mathbb{R}, |\cdot|) \\
{\scriptstyle \rho}\big\uparrow & & \big\uparrow{\scriptstyle f} \\
(V, \|\cdot\|_{\mathcal{H}}) & \xleftarrow{\quad I \quad} & (W, \|\cdot\|_{\ell_2})
\end{array}
\qquad (7)
$$

Since both $\rho$ and $I$ are bijective, it is easy to see that $f$ not only defines a function $g : X \to \mathbb{R}$ by $g := f \circ I \circ \rho$, but conversely, every function $g : X \to \mathbb{R}$ has such a representation and this representation is unique. In other words, there is a one-to-one relationship between the functions $X \to \mathbb{R}$ and the functions $W \to \mathbb{R}$. Let us now assume that we have a kernel $k_W$ on $W$ with RKHS $H_W$ and canonical feature map $\Phi_W : W \to H_W$. Then $k_X : X \times X \to \mathbb{R}$, given by

$$k_X(x, x') := k_W(I \circ \rho(x), I \circ \rho(x')), \qquad x, x' \in X,$$

defines a kernel on $X$, since

$$k_X(x, x') = k_W(I \circ \rho(x), I \circ \rho(x')) = \langle \Phi_W(I(\rho(x'))), \Phi_W(I(\rho(x)))\rangle_{H_W}, \qquad x, x' \in X,$$

shows that $\Phi_W \circ I \circ \rho : X \to H_W$ is a feature map of $k_X$. Moreover, [31, Theorem 4.21] shows that the RKHS $H_X$ of $k_X$ is given by

$$H_X = \left\{ \langle f, \Phi_W \circ I \circ \rho(\,\cdot\,) \rangle_{H_W} : f \in H_W \right\}.$$

Since, for $f \in H_W$, the reproducing property of $H_W$ gives $f \circ I \circ \rho(x) = \langle f, \Phi_W \circ I \circ \rho(x) \rangle_{H_W}$ for all $x \in X$ we thus conclude that $H_X = \{ f \circ I \circ \rho : f \in H_W \} =: H_W \circ I \circ \rho$. Let us now assume that $X$ is compact and that $k_W$ is one of the universal kernels considered in Theorem 2.1 or the Gaussian RBF kernel (4). Then the proof of Theorem 2.2 shows that $k_X$ is one of the universal kernels considered in Theorem 2.2. Moreover, if we consider the kernel $k_V : V \times V \to \mathbb{R}$ defined by $k_V(v,v') := k_W(I(v), I(v'))$, then an analogous argument shows that $k_V$ is a universal kernel. This raises the question, whether we need the compactness of $X$, or whether it suffices to assume that $\rho$ is injective, continuous and has a compact image $V$. Surprisingly, the answer is that it depends on the type of universality one needs. Indeed, if $\rho$ is as in Theorem 2.2, then the compactness of $X$ ensures that $\rho$ is a homeomorphism, that is, $\rho^{-1} : V \to X$ is continuous, too. Since $I$ is clearly also a homeomorphism, we can easily conclude that $C(X) = C(W) \circ I \circ \rho$, that is, we have the same relationship as we have for the RKHSs $H_W$ and $H_X$. From this, the universality is easy to establish. Let us now assume the compactness of $V$ instead of the compactness of $X$. Then, in general, $\rho$ is not a homeomorphism and the sets of continuous functions on $X$ and $V$ are in general different, even if we consider the set of bounded continuous functions on $X$. To see the latter, consider e.g. the map $\rho : [0,1) \to S^1$ onto the unit sphere $S^1$ of $\mathbb{R}^2$ defined by $\rho(t) := (\sin(2\pi t), \cos(2\pi t))$. Now this difference makes it impossible to conclude from the universality of $k_V$ (or $k_W$) to the universality of $k_X$. However, if $\tau_V$ denotes the topology of $V$, then $\rho^{-1}(\tau_V) := \{\rho^{-1}(O) : O \in \tau_V\}$ defines a new topology on $X$, which satisfies $\rho^{-1}(\tau_V) \subset \tau_X$. Consequently, there are, in general, fewer continuous functions with respect to $\rho^{-1}(\tau_V)$. Now, it is easy to check that $d_\rho(x,x') := \|\rho(x) - \rho(x')\|_{\mathcal{H}}$ defines a metric that generates $\rho^{-1}(\tau_V)$ and, since $\rho$ is isometric with respect to this new metric, we can conclude that $(X, d_\rho)$ is a compact metric space. Consequently, we are back in the situation of Theorem 2.2, and hence $k_X$ is universal with respect to the space $C(X, d_\rho)$ of functions $X \to \mathbb{R}$ that are continuous with respect to $d_\rho$. In other words, while $H_X$ may fail to approximate every function that is continuous with respect to $d_X$, it does approximate every function that is continuous with respect to $d_\rho$. Whether the latter approximation property is enough clearly depends on the specific application at hand.

Let us now present some universal kernels of practical interest. Please note, that although the function $\rho$ in our examples is even linear, the Theorem 2.2 only assumes $\rho$ to be continuous and injective. We start with two examples where $X$ is the set of distributions on some space $\Omega$.

**Example 1: universal kernels on the set of probability measures.**
Let $(\Omega, d_\Omega)$ be a compact metric space, $\mathcal{B}(\Omega)$ be its Borel $\sigma$-algebra, and $X := \mathcal{M}_1(\Omega)$ be the set of all Borel probability measures on $\Omega$. Then the topology describing weak convergence of probability measures can be metrized, e.g., by the Prohorov metric

$$d_X(\mathrm{P},\mathrm{P}') := \inf\{\varepsilon > 0 : \mathrm{P}(A) \le \mathrm{P}'(A^\varepsilon) + \varepsilon \text{ for all } A \in \mathcal{B}(\Omega)\}, \qquad \mathrm{P},\mathrm{P}' \in X, \qquad (8)$$

where $A^\varepsilon := \{\omega' \in \Omega : d_\Omega(\omega, \omega') < \varepsilon \text{ for some } \omega \in A\}$, see e.g. [2, Theorem 6.8, p. 73]. Moreover, $(X, d_X)$ is a compact metric space if and only if $(\Omega, d_\Omega)$ is a compact metric space, see [19, Thm. 6.4]. In order to construct universal kernels on $(X, d_X)$ with the help of Theorem 2.2, it thus remains to find separable Hilbert spaces $\mathcal{H}$ and injective, continuous embeddings $\rho : X \to \mathcal{H}$. Let $k_\Omega$ be a continuous kernel on $\Omega$ with RKHS $\mathcal{H}_\Omega$ and canonical feature map $\Phi_\Omega(\omega) := k_\Omega(\omega, \cdot)$, $\omega \in \Omega$. Note that $k_\Omega$ is bounded because it is continuous and $\Omega$ is compact. Then $\mathcal{H}_\Omega$ is separable and $\Phi_\Omega$ is bounded and continuous, see [31, Lemmata 4.23, 4.29, 4.33]. Assume that $k_\Omega$ is additionally *characteristic*, i.e. the function $\rho : X \to \mathcal{H}_\Omega$ defined by the Bochner integral $\rho(\mathrm{P}) := \mathbb{E}_\mathrm{P}\Phi_\Omega$ is *injective*. Then the next lemma, which is taken from [10, Thm. 5.1] and which is a modification of a theorem in [3, p. III. 40], ensures the continuity of $\rho$.

**Lemma 2.3** *Let $(\Omega, d_\Omega)$ be a complete separable metric space, $\mathcal{H}$ be a separable Banach space and $\Phi : \Omega \to \mathcal{H}$ be a bounded, continuous function. Then $\rho : \mathcal{M}_1(\Omega) \to \mathcal{H}$ defined by $\rho(\mathrm{P}) := \mathbb{E}_\mathrm{P}\Phi$ is continuous, i.e., $\mathbb{E}_{\mathrm{P}_n}\Phi \to \mathbb{E}_\mathrm{P}\Phi$, whenever $(\mathrm{P}_n)_{n \in \mathbb{N}} \subset \mathcal{M}_1(\Omega)$ converges weakly in $\mathcal{M}_1(\Omega)$ to $\mathrm{P}$.*

Consequently, the map $\rho : \mathcal{M}_1(\Omega) \to \mathcal{H}_\Omega$ satisfies the assumptions of Theorem 2.2, and hence the Gaussian-type RBF kernel

$$k_\sigma(\mathrm{P},\mathrm{P}') := \exp\left(-\sigma^2 \|\mathbb{E}_\mathrm{P}\Phi_\Omega - \mathbb{E}_{\mathrm{P}'}\Phi_\Omega\|^2_{\mathcal{H}_\Omega}\right), \quad \mathrm{P},\mathrm{P}' \in \mathcal{M}_1(\Omega), \qquad (9)$$

is *universal* and obviously bounded. Note that this kernel is conceptionally different to characteristic kernels on $\Omega$. Indeed, characteristic kernels live on $\Omega$ and their RKHS consist of functions $\Omega \to \mathbb{R}$, while the new kernel $k_\sigma$ lives on $\mathcal{M}_1(\Omega)$ and its RKHS consists of functions $\mathcal{M}_1(\Omega) \to \mathbb{R}$. Consequently, $k_\sigma$ can be used to *learn* from samples that are individual distributions, e.g. represented by histograms, densities or data, while characteristic kernels can only be used to check whether two of such distributions are equal or not.

**Example 2: universal kernels based on Fourier transforms of probability measures.**
Consider, the set $X := \mathcal{M}_1(\Omega)$, where $\Omega \subset \mathbb{R}^d$ is compact. Moreover, let $\rho$ be the *Fourier transform* (or *characteristic function*), that is $\rho(\mathrm{P}) := \hat{\mathrm{P}}$, where $\hat{\mathrm{P}}(t) := \int e^{i\langle z,t\rangle} d\mu(z) \in \mathbb{C}$, $t \in \mathbb{R}^d$. It is well-known, see e.g. [6, Chap. 9], that, for all $\mathrm{P} \in \mathcal{M}_1(\Omega)$, $\hat{\mathrm{P}}$ is uniformly continuous on $\mathbb{R}^d$ and $\|\hat{\mathrm{P}}\|_\infty \leq 1$. Moreover, $\rho : \mathrm{P} \mapsto \hat{\mathrm{P}}$ is injective, and if a sequence $(\mathrm{P}_n)$ converges weakly to some P, then $(\hat{\mathrm{P}}_n)$ converges uniformly to $\hat{\mathrm{P}}$ on every compact subset of $\mathbb{R}^d$. Now let $\mu$ be a finite Borel measure on $\mathbb{R}^d$ with support$(\mu) = \mathbb{R}^d$, e.g., $\mu$ can be any probability distribution on $\mathbb{R}^d$ with Lebesgue density $h > 0$. Then the previous properties of the Fourier transform can be used to show that $\rho : \mathcal{M}_1(\Omega) \to L_2(\mu)$ is continuous, and hence Theorem 2.2 ensures that the following Gaussian-type kernel is *universal* and bounded:

$$k_\sigma(\mathrm{P}, \mathrm{P}') := \exp\left(-\sigma^2 \|\hat{\mathrm{P}} - \hat{\mathrm{P}}'\|_{L_2(\mu)}^2\right), \qquad \mathrm{P}, \mathrm{P}' \in \mathcal{M}_1(\Omega). \tag{10}$$

In view of the previous two examples, we mention that the probability measures P and P′ are often not directly observable in practice, but only corresponding empirical distributions can be obtained. In this case, a simple standard technique is to construct histograms to represent these empirical distributions as vectors in a finite-dimensional Euclidean space, although it is well-known that histograms can yield bad estimates for probability measures. Our new kernels make it possible to directly plug the empirical distributions into the kernel $k_\sigma$, even if these distributions do not have the same length. Moreover, other techniques to convert empirical distributions to absolutely continuous distributions such as kernel estimators derived via weighted averaging of rounded points (WAPRing) and (averaging) histograms with different origins, [20, 24] can be used in $k_\sigma$, too. Clearly, the preferred method will most likely depend on the specific application at hand, and one benefit of our construction is that it allows this flexibility.

**Example 3: universal kernels for signal processing.**
Let $(\Omega, \mathcal{A}, \mu)$ be an arbitrary measure space and $L_2(\mu)$ be the usual space of square $\mu$-integrable functions on $\Omega$. Let us additionally assume that $L_2(\mu)$ is separable, which is typically, but not always, satisfied. In addition, let us assume that our input values $x_i \in X$ are functions taken from some compact set $X \subset L_2(\mu)$. A typical example, where this situation occurs, is signal processing, where the true signal $f \in L_2([0,1])$, which is a function of time, cannot be directly observed, but a smoothed version $g := T \circ f$ of the signal is observable. This smoothing can often be described by a compact linear operator $T : L_2([0,1]) \to L_2([0,1])$, e.g., a convolution operator, acting on the true signals. Hence, if we assume that the true signals are contained in the closed unit ball $B_{L_2([0,1])}$, then the observed, smoothed signals $T \circ f$ are contained in a compact subset $X$ of $L_2([0,1])$. Let us now return to the general case introduced above. Then the identity map $\rho := \mathrm{id} : X \to L_2(\mu)$ satisfies the assumptions of Theorem 2.2, and hence the Gaussian-type kernel

$$k_\sigma(g, g') := \exp\left(-\sigma^2 \|g - g'\|_{L_2(\mu)}^2\right), \qquad g, g' \in X, \tag{11}$$

defines a universal and bounded kernel on $X$. As in the previous examples, note that the computation of $k_\sigma$ does not require the functions $g$ and $g'$ to be in a specific format such as a certain discretization.

## 3  Discussion

The main goal of this paper was to provide an explicit construction of universal kernels that are defined on arbitrary compact metric spaces, which are not necessarily a subset of $\mathbb{R}^d$. There is a still increasing interest in kernel methods including support vector machines on such input spaces, e.g. for classification or regression purposes for input values being probability measures, histograms or colored images. As examples, we gave explicit universal kernels on the set of probability distributions and for signal processing. One direction of further research may be to generalize our results to the case of non-compact metric spaces or to find quantitative approximation results.

# 4 Proofs

In the following, we write $\mathbb{N}_0^{\mathbb{N}}$ for the set of all sequences $(j_i)_{i \geq 1}$ with values in $\mathbb{N}_0 := \mathbb{N} \cup \{0\}$. Elements of this set will serve us as multi-indices with countably many components. For $j = (j_i) \in \mathbb{N}_0^{\mathbb{N}}$, we will therefore adopt the multi-index notation

$$|j| := \sum_{i \geq 1} j_i \,.$$

Note that $|j| < \infty$ implies that $j$ has only finitely many components $j_i$ with $j_i \neq 0$.

**Lemma 4.1** *Assume that $n \in \mathbb{N}$ is fixed and that for all $j \in \mathbb{N}_0^{\mathbb{N}}$ with $|j| = n$, we have some constant $c_j \in (0, \infty)$. Then for all $j \in \mathbb{N}_0^{\mathbb{N}}$ with $|j| = n + 1$, there exists a constant $\tilde{c}_j \in (0, \infty)$ such that for all summable sequences $(b_i) \subset [0, \infty)$ we have*

$$\left( \sum_{j \in \mathbb{N}_0^{\mathbb{N}} : |j| = n} c_j \prod_{i=1}^{\infty} b_i^{j_i} \right) \left( \sum_{i=1}^{\infty} b_i \right) = \sum_{j \in \mathbb{N}_0^{\mathbb{N}} : |j| = n+1} \tilde{c}_j \prod_{i=1}^{\infty} b_i^{j_i} \,.$$

**Proof:** This can be shown by induction, where the induction step is similar to the proof for the Cauchy product of series. ∎

**Lemma 4.2** *Assume that $n \in \mathbb{N}_0$ is fixed. Then for all $j \in \mathbb{N}_0^{\mathbb{N}}$ with $|j| = n$, there exists a constant $c_j \in (0, \infty)$ such that for all summable sequences $(b_i) \subset [0, \infty)$ we have*

$$\left( \sum_{i=1}^{\infty} b_i \right)^n = \sum_{j \in \mathbb{N}_0^{\mathbb{N}} : |j| = n} c_j \prod_{i=1}^{\infty} b_i^{j_i} \,.$$

**Proof:** This can be shown by induction using Lemma 4.1. ∎

Given a non-empty countable set $J$ and a family $w := (w_j)_{j \in J} \subset \mathbb{R}$, we write $\|w\|_2^2 := \sum_{j \in J} w_j^2$, and, as usual, we denote the space of all families for which this quantity is finite by $\ell_2(J)$. Recall that $\ell_2(J)$ together with $\| \cdot \|_2$ is a Hilbert space and we denote its inner product by $\langle \cdot, \cdot \rangle_{\ell_2(J)}$. Moreover, $\ell_2 := \ell_2(\mathbb{N})$ is separable, and by using an orthonormal basis representation, it is further known that every separable Hilbert space is isometrically isomorphic to $\ell_2$. In this sense, $\ell_2$ can be viewed as a generic model for separable Hilbert spaces.

The following result provides a method to construct Taylor kernels on closed balls in $\ell_2$.

**Proposition 4.3** *Let $r \in (0, \infty]$ and $K : [-r, r] \to \mathbb{R}$ be a function that can be expressed by its Taylor series given in (1), i.e. $K(t) = \sum_{n=0}^{\infty} a_n t^n$, $t \in [-r, r]$. Define $J := \{j \in \mathbb{N}_0^{\mathbb{N}} : |j| < \infty\}$. If $a_n \geq 0$ for all $n \geq 0$, then $k : \sqrt{r} B_{\ell_2} \times \sqrt{r} B_{\ell_2} \to \mathbb{R}$ defined by (3), i.e.*

$$k(w, w') := K\big( \langle w, w' \rangle_{\ell_2} \big) = \sum_{n=0}^{\infty} a_n \langle w, w' \rangle_{\ell_2}^n, \qquad w, w' \in \sqrt{r} B_{\ell_2},$$

*is a kernel. Moreover, for all $j \in J$, there exists a constant $c_j \in (0, \infty)$ such that $\Phi : \sqrt{r} B_{\ell_2} \to \ell_2(J)$ defined by*

$$\Phi(w) := \left( c_j \prod_{i=1}^{\infty} w_i^{j_i} \right)_{j \in J}, \qquad w \in \sqrt{r} B_{\ell_2}, \tag{12}$$

*is a feature map of $k$, where we use the convention $0^0 := 1$.*

**Proof:** For $w, w' \in \sqrt{r} B_{\ell_2}$, the Cauchy-Schwarz inequality yields $|\langle w, w' \rangle| \leq \|w\|_2 \|w'\|_2 \leq r$ and thus $k$ is well-defined. Let $w_i$ denote the $i$-th component of $w \in \ell_2$. Since (1) is absolutely convergent, Lemma 4.2 then shows that, for all $j \in \mathbb{N}_0^{\mathbb{N}}$, there exists a constant $\tilde{c}_j \in (0, \infty)$ such that

$$k(w, w') = \sum_{j \in \mathbb{N}_0^{\mathbb{N}}} a_{|j|} \tilde{c}_j \prod_{i=1}^{\infty} (w_i')^{j_i} \prod_{i=1}^{\infty} w_i^{j_i} \,.$$

Setting $c_j := \sqrt{a_{|j|} \tilde{c}_j}$, we obtain that $\Phi$ defined in (12) is indeed a feature map of $k$, and hence $k$ is a kernel. ∎

Before we can state our first main result the need to recall the following test of universality from [31, Theorem 4.56].

**Theorem 4.4** *Let $W$ be a compact metric space and $k$ be a continuous kernel on $W$ with $k(w, w) > 0$ for all $w \in W$. Suppose that we have an injective feature map $\Phi : W \to \ell_2(J)$ of $k$, where $J$ is some countable set. We write $\Phi_j : W \to \mathbb{R}$ for its $j$-th component, i.e., $\Phi(w) = (\Phi_j(w))_{j \in J}$, $w \in W$. If $\mathcal{A} := \operatorname{span} \{\Phi_j : j \in J\}$ is an algebra, then $k$ is universal.*

With the help of Theorem 4.4 and Proposition 4.3 we can now prove our first main result.

***Proof of Theorem 2.1:*** We have already seen in Proposition 4.3 that $k$ is a kernel on $\sqrt{r}B_{\ell_2}$. Let us now fix a compact $W \subset \sqrt{r}B_{\ell_2}$. For every $j \in J$, where $J$ is defined in Proposition 4.3, there are only finitely many components $j_i$ with $j_i \neq 0$. Consequently, there exists a bijection between $J$ and the set of all finite subsets of $\mathbb{N}$. Since the latter is countable, $J$ is countable. Furthermore, we have

$$k(w, w) = \sum_{n=0}^{\infty} a_n \|w\|_{\ell_2}^{2n} \geq a_0 > 0$$

for all $w \in W$, and it is obvious, that the components of the feature map $\Phi$ found in Proposition 4.3 span an algebra. Finally, if we have $w, w' \in W$ with $w \neq w'$, there exists an $i \geq 1$ such that $w_i \neq w_i'$. For the multi-index $j \in J$ that equals 1 at the $i$-component and vanishes everywhere else we then have $\Phi(w) = c_j w_i \neq c_j w_i' = \Phi(w')$, and hence $\Phi$ is injective. ∎

***Proof of Theorem 2.2:*** Since $\mathcal{H}$ is separable Hilbert space there exists an isometric isomorphism $I : \mathcal{H} \to \ell_2$. We define $V := \rho(X)$, see also the diagram in (7). Since $\rho$ is continuous, $V$ is the image of a compact set under a continuous map, and thus $V$ is compact and the inverse of the bijective map $I \circ \rho : X \to W$ is continuous. Consequently, there is a one-to-one relationship between the continuous functions $f_X$ on $X$ and the continuous functions $f_W$ on $W$, namely $C(X) = C(W) \circ I \circ \rho$, see also the discussion following (7). Moreover, the fact that $I : \mathcal{H} \to \ell_2$ is an isometric isomorphism yields $\langle I(\rho(x)), I(\rho(x')) \rangle_{\ell_2} = \langle \rho(x), \rho(x') \rangle_{\mathcal{H}}$ for all $x, x' \in X$, and hence the kernel $k$ considered in Theorem 2.2 is of the form $k_X = k_W(I \circ \rho(\,\cdot\,), I \circ \rho(\,\cdot\,))$, where $k_W$ is the corresponding kernel defined on $W \subset \ell_2$ considered in Theorem 2.2. Now, the discussion following (7) showed $H_X = H_W \circ I \circ \rho$. Consequently, if we fix a function $g \in C(X)$, then $f := g \circ \rho^{-1} \circ I^{-1} \in C(W)$ can be approximated by $H_W$, that is, for all $\varepsilon > 0$, there exists an $h \in H_W$ such that $\|h - f\|_\infty \leq \varepsilon$. Since $I \circ \rho : X \to W$ is bijective and $f \circ I \circ \rho = g$, we conclude that $\|h \circ I \circ \rho - g\|_\infty \leq \varepsilon$. Now the assertion follows from $h \circ I \circ \rho \in H_X$. ∎

## References

[1] H. Bauer. *Measure and Integration Theory*. De Gruyter, Berlin, 2001.

[2] P. Billingsley. *Convergence of probability measures*. John Wiley & Sons, New York, 2nd edition, 1999.

[3] N. Bourbaki. *Integration I. Chapters 1-6*. Springer, Berlin, 2004. Translated from the 1959, 1965, and 1967 French originals by S.K. Berberian.

[4] A. Caponnetto, C.A. Micchelli, M. Pontil, and Y. Ying. Universal multi-task kernels. *J. Mach. Learn. Res.*, 9:1615–1646, 2008.

[5] O. Chapelle, P. Haffner, and V. Vapnik. SVMs for histogram-based image classification. *IEEE Transactions on Neural Networks*, 10:1055–1064, 1999.

[6] R. M. Dudley. *Real Analysis and Probability*. Cambridge University Press, Cambridge, 2002.

[7] K. Fukumizu, F. R. Bach, and M. I. Jordan. Dimensionality reduction for supervised learning with reproducing kernel hilbert spaces. *J. Mach. Learn. Res.*, 5:73–99, 2005.

[8] K. Fukumizu, F. R. Bach, and M. I. Jordan. Kernel Dimension Reduction in Regression. *Ann. Statist.*, 37:1871–1905, 2009.

[9] K. Fukumizu, B. K. Sriperumbudur, A. Gretton, and B. Schölkopf. Characteristic kernels on groups and semigroups. In D. Koller, D. Schuurmans, Y. Bengio, and L. Bottou, editors, *Advances in Neural Information Processing Systems 21*, pages 473–480. 2009.

[10] R. Hable and A. Christmann. Qualitative robustness of support vector machines. *arXiv:0912.0874v1*, 2009.

[11] M. Hein and O. Bousquet. Kernels, associated structures and generalizations. Technical report, Max-Planck-Institute for Biological Cybernetics, 2004.

[12] M. Hein and O. Bousquet. Hilbertian metrics and positive definite kernels on probability measures. In Z. Ghahramani and R. Cowell, editors, *AISTATS*, pages 136–143, 2005.

[13] M. Hein, O. Bousquet, and B. Schölkopf. Maximal margin classification for metric spaces. *Journal of Computer and System Sciences*, 71:333–359, 2005.

[14] M. Hein, T. N. Lal, and O. Bousquet. Hilbertian metrics on probability measures and their application in SVM's. In C. E. Rasmussen, H. H. Bülthoff, M. Giese, and B. Schölkopf, editors, *Pattern Recognition, Proceedings of the 26th DAGM Symposium*, pages 270–277, Berlin, 2004. Springer.

[15] T. Joachims. *Learning to Classify Text Using Support Vector Machines*. Kluwer Academic Publishers, Boston, 2002.

[16] J. Lafferty and G. Lebanon. Diffusion kernels on statistical manifolds. *J. Mach. Learn. Res.*, 6:129–163, 2005.

[17] A.F.T. Martins, N.A. Smith, E.P. Xing, P.M.Q. Aguiar, and M.A.T. Figueiredo. Nonextensive information theoretic kernels on measures. *J. Mach. Learn. Res.*, 10:935–975, 2009.

[18] C. A. Micchelli, Y. Xu, and H. Zhang. Universal kernels. *J. Mach. Learn. Res.*, 7:2651–2667, 2006.

[19] K. R. Parthasarathy. *Probability Measures on Metric Spaces*. Academic Press, New York, 1967.

[20] E. Parzen. On estimating of a probability density and mode. *Ann. Math. Statist.*, 35:1065–1076, 1962.

[21] A. Pinkus. Strictly positive definite functions on a real inner product space. *Adv. Comput. Math.*, 20:263–271, 2004.

[22] B. Schölkopf, A. J. Smola, and K.-R. Müller. Nonlinear component analysis as a kernel eigenvalue problem. *Neural Comput.*, 10:1299–1319, 1998.

[23] B. Schölkopf, K. Tsuda, and J. P. Vert. *Kernel Methods in Computational Biology*. MIT Press, Cambridge, MA, 2004.

[24] D. Scott. Averaged shifted histograms: Effective nonparametric density estimation in several dimensions. *Ann. Statist.*, 13:1024–1040, 1985.

[25] C. Scovel, D. Hush, I. Steinwart, and J. Theiler. Radial kernels and their reproducing kernel Hilbert spaces. *Journal of Complexity*, 2010, to appear.

[26] A.J. Smola, A. Gretton, L. Song, and B. Schölkopf. A Hilbert Space Embedding for Distributions. In E. Takimoto, editor, *Algorithmic Learning Theory*, Lecture Notes on Computer Science. Springer, 2007. Proceedings of the 10th International Conference on Discovery Science, 40-41.

[27] B. Sriperumbudur, K. Fukumizu, A. Gretton, G. Lanckriet, and B. Schölkopf. Kernel choice and classifiability for RKHS embeddings of probability distributions. In Y. Bengio, D. Schuurmans, J. Lafferty, C. K. I. Williams, and A. Culotta, editors, *Advances in Neural Information Processing Systems 22*, pages 1750–1758. 2009.

[28] B. Sriperumbudur, K. Fukumizu, and G. Lanckriet. On the relation between universality, characteristic kernels and RKHS embeddings of measures. In Yee Whye Teh and M. Titterington, editors, *AISTATS 2010, Proc. of the 13th International Conference on Artificial Intelligence and Statistics*, volume 9, pages 773–780. 2010.

[29] B. Sriperumbudur, K. Fukumizu, and G. Lanckriet. Universality, characteristic kernels and RKHS embeddings of measures. *arXiv:1003.0887v1*, 2010.

[30] I. Steinwart. On the influence of the kernel on the consistency of support vector machines. *J. Mach. Learn. Res.*, 2:67–93, 2001.

[31] I. Steinwart and A. Christmann. *Support Vector Machines*. Springer, New York, 2008.

[32] I. Steinwart, D. Hush, and C. Scovel. Function classes that approximate the Bayes risk. In *COLT'06, 19th Conference on Learning Theory*, Pittsburgh, 2006.

